# Kernel PCA and De-Noising in Feature Spaces

**Sebastian Mika, Bernhard Schölkopf, Alex Smola**
**Klaus-Robert Müller, Matthias Scholz, Gunnar Rätsch**
GMD FIRST, Rudower Chaussee 5, 12489 Berlin, Germany
{mika, bs, smola, klaus, scholz, raetsch}@first.gmd.de

## Abstract

Kernel PCA as a nonlinear feature extractor has proven powerful as a preprocessing step for classification algorithms. But it can also be considered as a natural generalization of linear principal component analysis. This gives rise to the question how to use nonlinear features for data compression, reconstruction, and de-noising, applications common in linear PCA. This is a nontrivial task, as the results provided by kernel PCA live in some high dimensional feature space and need not have pre-images in input space. This work presents ideas for finding approximate pre-images, focusing on Gaussian kernels, and shows experimental results using these pre-images in data reconstruction and de-noising on toy examples as well as on real world data.

## 1 PCA and Feature Spaces

Principal Component Analysis (PCA) (e.g. [3]) is an orthogonal basis transformation. The new basis is found by diagonalizing the centered covariance matrix of a data set $\{x_k \in \mathbf{R}^N | k = 1, \dots, \ell\}$, defined by $C = \langle (x_i - \langle x_k \rangle)(x_i - \langle x_k \rangle)^T \rangle$. The coordinates in the Eigenvector basis are called *principal components*. The size of an Eigenvalue $\lambda$ corresponding to an Eigenvector $v$ of $C$ equals the amount of variance in the direction of $v$. Furthermore, the directions of the first $n$ Eigenvectors corresponding to the biggest $n$ Eigenvalues cover as much variance as possible by $n$ orthogonal directions. In many applications they contain the most interesting information: for instance, in data compression, where we project onto the directions with biggest variance to retain as much information as possible, or in de-noising, where we deliberately drop directions with small variance.

Clearly, one cannot assert that linear PCA will always detect all structure in a given data set. By the use of suitable *nonlinear* features, one can extract more information. Kernel PCA is very well suited to extract interesting nonlinear structures in the data [9]. The purpose of this work is therefore (i) to consider nonlinear de-noising based on Kernel PCA and (ii) to clarify the connection between feature space expansions and meaningful patterns in input space. Kernel PCA first maps the data into some feature space **F** via a (usually nonlinear) function $\Phi$ and then performs linear PCA on the mapped data. As the feature space **F** might be very high dimensional (e.g. when mapping into the space of all possible $d$-th order monomials of input space), kernel PCA employs Mercer kernels instead of carrying

out the mapping $\Phi$ explicitly. A Mercer kernel is a function $k(\boldsymbol{x}, \boldsymbol{y})$ which for all data sets $\{\boldsymbol{x}_i\}$ gives rise to a positive matrix $K_{ij} = k(\boldsymbol{x}_i, \boldsymbol{x}_j)$ [6]. One can show that using k instead of a dot product in input space corresponds to mapping the data with some $\Phi$ to a feature space **F** [1], i.e. $k(\boldsymbol{x}, \boldsymbol{y}) = (\Phi(\boldsymbol{x}) \cdot \Phi(\boldsymbol{y}))$. Kernels that have proven useful include Gaussian kernels $k(\boldsymbol{x}, \boldsymbol{y}) = \exp(-\|\boldsymbol{x} - \boldsymbol{y}\|^2/c)$ and polynomial kernels $k(\boldsymbol{x}, \boldsymbol{y}) = (\boldsymbol{x} \cdot \boldsymbol{y})^d$. Clearly, all algorithms that can be formulated in terms of dot products, e.g. *Support Vector Machines* [1], can be carried out in some feature space **F** without mapping the data explicitly. All these algorithms construct their solutions as expansions in the potentially infinite-dimensional feature space.

The paper is organized as follows: in the next section, we briefly describe the kernel PCA algorithm. In section 3, we present an algorithm for finding approximate pre-images of expansions in feature space. Experimental results on toy and real world data are given in section 4, followed by a discussion of our findings (section 5).

## 2   Kernel PCA and Reconstruction

To perform PCA in feature space, we need to find Eigenvalues $\lambda > 0$ and Eigenvectors $V \in \mathbf{F}\backslash\{0\}$ satisfying $\lambda V = \bar{C}V$ with $\bar{C} = \langle \Phi(\boldsymbol{x}_k)\Phi(\boldsymbol{x}_k)^T \rangle$.[1] Substituting $\bar{C}$ into the Eigenvector equation, we note that all solutions $V$ must lie in the span of $\Phi$-images of the training data. This implies that we can consider the equivalent system

$$\lambda(\Phi(\boldsymbol{x}_k) \cdot V) = (\Phi(\boldsymbol{x}_k) \cdot \bar{C}V) \text{ for all } k = 1, \ldots, \ell \tag{1}$$

and that there exist coefficients $\alpha_1, \ldots, \alpha_\ell$ such that

$$V = \sum_{i=1}^{\ell} \alpha_i \Phi(\boldsymbol{x}_i) \tag{2}$$

Substituting $\bar{C}$ and (2) into (1), and defining an $\ell \times \ell$ matrix $K$ by $K_{ij} := (\Phi(\boldsymbol{x}_i) \cdot \Phi(\boldsymbol{x}_j)) = k(\boldsymbol{x}_i, \boldsymbol{x}_j)$, we arrive at a problem which is cast in terms of dot products: solve

$$\ell\lambda\alpha = K\alpha \tag{3}$$

where $\alpha = (\alpha_1, \ldots, \alpha_\ell)^T$ (for details see [7]). Normalizing the solutions $V^k$, i.e. $(V^k \cdot V^k) = 1$, translates into $\lambda_k(\alpha^k \cdot \alpha^k) = 1$. To extract nonlinear principal components for the $\Phi$-image of a test point $\boldsymbol{x}$ we compute the projection onto the $k$-th component by $\beta_k := (V^k \cdot \Phi(\boldsymbol{x})) = \sum_{i=1}^{\ell} \alpha_i^k k(\boldsymbol{x}, \boldsymbol{x}_i)$. For feature extraction, we thus have to evaluate $\ell$ kernel functions instead of a dot product in **F**, which is expensive if **F** is high-dimensional (or, as for Gaussian kernels, infinite-dimensional). To reconstruct the $\Phi$-image of a vector $\boldsymbol{x}$ from its projections $\beta_k$ onto the first $n$ principal components in **F** (assuming that the Eigenvectors are ordered by decreasing Eigenvalue size), we define a projection operator $P_n$ by

$$P_n\Phi(\boldsymbol{x}) = \sum_{k=1}^{n} \beta_k V^k \tag{4}$$

If $n$ is large enough to take into account all directions belonging to Eigenvectors with non-zero Eigenvalue, we have $P_n\Phi(\boldsymbol{x}_i) = \Phi(\boldsymbol{x}_i)$. Otherwise (kernel) PCA still satisfies (i) that the overall squared reconstruction error $\sum_i \| P_n\Phi(\boldsymbol{x}_i) - \Phi(\boldsymbol{x}_i)\|^2$ is minimal and (ii) the retained variance is maximal among all projections onto orthogonal directions in **F**. In common applications, however, we are interested in a reconstruction in input space rather than in **F**. The present work attempts to achieve this by computing a vector $\boldsymbol{z}$ satisfying $\Phi(\boldsymbol{z}) = P_n\Phi(\boldsymbol{x})$. The hope is that for the kernel used, such a $\boldsymbol{z}$ will be a good approximation of $\boldsymbol{x}$ in input space. However, (i) such a $\boldsymbol{z}$ will not always exist and (ii) if it exists,

it need be not unique.[2] As an example for (i), we consider a possible representation of $\mathbf{F}$. One can show [7] that $\Phi$ can be thought of as a map $\Phi(x) = k(x, .)$ into a Hilbert space $\mathcal{H}_k$ of functions $\sum_i \alpha_i k(x_i, .)$ with a dot product satisfying $(k(x, .) \cdot k(y, .)) = k(x, y)$. Then $\mathcal{H}_k$ is called *reproducing kernel Hilbert space* (e.g. [6]). Now, for a Gaussian kernel, $\mathcal{H}_k$ contains all linear superpositions of Gaussian bumps on $\mathbf{R}^N$ (plus limit points), whereas by definition of $\Phi$ only single bumps $k(x, .)$ have pre-images under $\Phi$. When the vector $P_n\Phi(x)$ has no pre-image $z$ we try to approximate it by minimizing

$$\rho(z) = \|\Phi(z) - P_n\Phi(x)\|^2 \tag{5}$$

This is a special case of the reduced set method [2]. Replacing terms independent of $z$ by $\Omega$, we obtain

$$\rho(z) = \|\Phi(z)\|^2 - 2(\Phi(z) \cdot P_n\Phi(x)) + \Omega \tag{6}$$

Substituting (4) and (2) into (6), we arrive at an expression which is written in terms of dot products. Consequently, we can introduce a kernel to obtain a formula for $\rho$ (and thus $\nabla_z \rho$) which does not rely on carrying out $\Phi$ explicitly

$$\rho(z) = k(z, z) - 2 \sum_{k=1}^{n} \beta_k \sum_{i=1}^{\ell} \alpha_i^k k(z, x_i) + \Omega \tag{7}$$

## 3  Pre-Images for Gaussian Kernels

To optimize (7) we employed standard gradient descent methods. If we restrict our attention to kernels of the form $k(x, y) = k(\|x - y\|^2)$ (and thus satisfying $k(x, x) \equiv$ const. for all $x$), an optimal $z$ can be determined as follows (cf. [8]): we deduce from (6) that we have to maximize

$$\rho(z) = (\Phi(z) \cdot P_n\Phi(x)) + \Omega' = \sum_{i=1}^{\ell} \gamma_i k(z, x_i) + \Omega' \tag{8}$$

where we set $\gamma_i = \sum_{k=1}^{n} \beta_k \alpha_i^k$ (for some $\Omega'$ independent of $z$). For an extremum, the gradient with respect to $z$ has to vanish: $\nabla_z \rho(z) = \sum_{i=1}^{\ell} \gamma_i k'(\|z - x_i\|^2)(z - x_i) = 0$. This leads to a necessary condition for the extremum: $z = \sum_i \delta_i x_i / \sum_j \delta_j$, with $\delta_i = \gamma_i k'(\|z - x_i\|^2)$. For a Gaussian kernel $k(x, y) = \exp(-\|x - y\|^2/c)$ we get

$$z = \frac{\sum_{i=1}^{\ell} \gamma_i \exp(-\|z - x_i\|^2/c) x_i}{\sum_{i=1}^{\ell} \gamma_i \exp(-\|z - x_i\|^2/c)}. \tag{9}$$

We note that the denominator equals $(\Phi(z) \cdot P_n\Phi(x))$ (cf. (8)). Making the assumption that $P_n\Phi(x) \neq 0$, we have $(\Phi(x) \cdot P_n\Phi(x)) = (P_n\Phi(x) \cdot P_n\Phi(x)) > 0$. As k is smooth, we conclude that there exists a neighborhood of the extremum of (8) in which the denominator of (9) is $\neq 0$. Thus we can devise an iteration scheme for $z$ by

$$z_{t+1} = \frac{\sum_{i=1}^{\ell} \gamma_i \exp(-\|z_t - x_i\|^2/c) x_i}{\sum_{i=1}^{\ell} \gamma_i \exp(-\|z_t - x_i\|^2/c)} \tag{10}$$

Numerical instabilities related to $(\Phi(z) \cdot P_n\Phi(x))$ being small can be dealt with by restarting the iteration with a different starting value. Furthermore we note that any fixed-point of (10) will be a linear combination of the kernel PCA training data $x_i$. If we regard (10) in the context of clustering we see that it resembles an iteration step for the estimation of

the center of a single Gaussian cluster. The weights or 'probabilities' $\gamma_i$ reflect the (anti-) correlation between the amount of $\Phi(x)$ in Eigenvector direction $V^k$ and the contribution of $\Phi(x_i)$ to this Eigenvector. So the 'cluster center' $z$ is drawn towards training patterns with positive $\gamma_i$ and pushed away from those with negative $\gamma_i$, i.e. for a fixed-point $z_\infty$ the influence of training patterns with smaller distance to $x$ will tend to be bigger.

## 4  Experiments

To test the feasibility of the proposed algorithm, we run several toy and real world experiments. They were performed using (10) and Gaussian kernels of the form $k(x, y) = \exp(-(\|x - y\|^2)/(nc))$ where $n$ equals the dimension of input space. We mainly focused on the application of *de-noising*, which differs from *reconstruction* by the fact that we are allowed to make use of the original test data as starting points in the iteration.

**Toy examples:** In the first experiment (table 1), we generated a data set from eleven Gaussians in $\mathbf{R}^{10}$ with zero mean and variance $\sigma^2$ in each component, by selecting from each source 100 points as a training set and 33 points for a test set (centers of the Gaussians randomly chosen in $[-1, 1]^{10}$). Then we applied kernel PCA to the training set and computed the projections $\beta_k$ of the points in the test set. With these, we carried out de-noising, yielding an approximate pre-image in $\mathbf{R}^{10}$ for each test point. This procedure was repeated for different numbers of components in reconstruction, and for different values of $\sigma$. For the kernel, we used $c = 2\sigma^2$. We compared the results provided by our algorithm to those of linear PCA via the mean squared distance of all de-noised test points to their corresponding center. Table 1 shows the *ratio* of these values; here and below, ratios larger than one indicate that kernel PCA performed better than linear PCA. For almost every choice of $n$ and $\sigma$, kernel PCA did better. Note that using all 10 components, linear PCA is just a basis transformation and hence cannot de-noise. The extreme superiority of kernel PCA for small $\sigma$ is due to the fact that all test points are in this case located close to the eleven spots in input space, and linear PCA has to cover them with less than ten directions. Kernel PCA moves each point to the correct source even when using only a small number of components.

| $\sigma$ | $n=1$ | 2 | 3 | 4 | 5 | 6 | 7 | 8 | 9 |
|---|---|---|---|---|---|---|---|---|---|
| 0.05 | 2058.42 | 1238.36 | 846.14 | 565.41 | 309.64 | 170.36 | 125.97 | 104.40 | 92.23 |
| 0.1 | 10.22 | 31.32 | 21.51 | 29.24 | 27.66 | 23.53 | 29.64 | 40.07 | 63.41 |
| 0.2 | 0.99 | 1.12 | 1.18 | 1.50 | 2.11 | 2.73 | 3.72 | 5.09 | 6.32 |
| 0.4 | 1.07 | 1.26 | 1.44 | 1.64 | 1.91 | 2.08 | 2.22 | 2.34 | 2.47 |
| 0.8 | 1.23 | 1.39 | 1.54 | 1.70 | 1.80 | 1.96 | 2.10 | 2.25 | 2.39 |

Table 1: De-noising Gaussians in $\mathbf{R}^{10}$ (see text). Performance ratios larger than one indicate how much better kernel PCA did, compared to linear PCA, for different choices of the Gaussians' std. dev. $\sigma$, and different numbers of components used in reconstruction.

To get some intuitive understanding in a low-dimensional case, figure 1 depicts the results of de-noising a half circle and a square in the plane, using kernel PCA, a *nonlinear autoencoder, principal curves*, and linear PCA. The principal curves algorithm [4] iteratively estimates a curve capturing the structure of the data. The data are projected to the closest point on a curve which the algorithm tries to construct such that each point is the average of all data points projecting onto it. It can be shown that the only straight lines satisfying the latter are principal components, so principal curves are a generalization of the latter. The algorithm uses a smoothing parameter which is annealed during the iteration. In the nonlinear autoencoder algorithm, a 'bottleneck' 5-layer network is trained to reproduce the input values as outputs (i.e. it is used in autoassociative mode). The hidden unit activations in the third layer form a lower-dimensional representation of the data, closely related to

PCA (see for instance [3]). Training is done by conjugate gradient descent. In all algorithms, parameter values were selected such that the best possible de-noising result was obtained. The figure shows that on the closed square problem, kernel PCA does (subjectively) best, followed by principal curves and the nonlinear autoencoder; linear PCA fails completely. However, note that all algorithms except for kernel PCA actually provide an explicit one-dimensional parameterization of the data, whereas kernel PCA only provides us with a means of mapping points to their de-noised versions (in this case, we used four kernel PCA features, and hence obtain a four-dimensional parameterization).

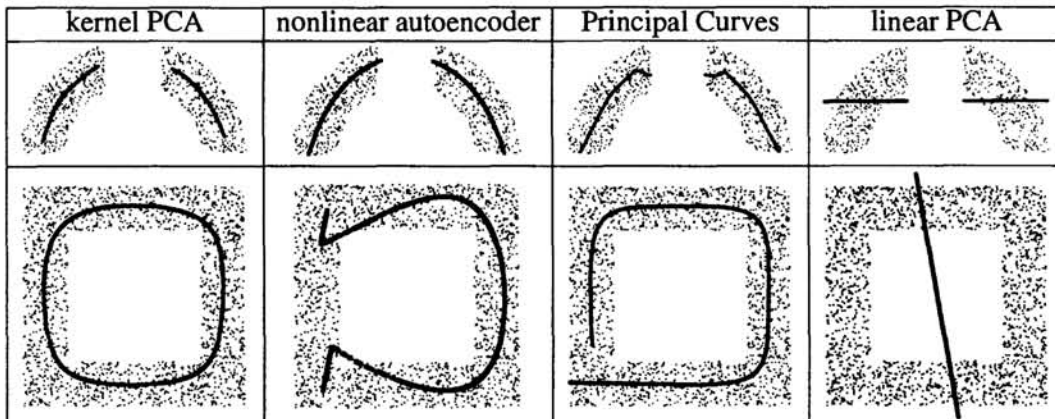

| kernel PCA | nonlinear autoencoder | Principal Curves | linear PCA |

Figure 1: De-noising in 2-d (see text). Depicted are the data set (small points) and its de-noised version (big points, joining up to solid lines). For linear PCA, we used one component for reconstruction, as using two components, reconstruction is perfect and thus does not de-noise. Note that all algorithms except for our approach have problems in capturing the circular structure in the bottom example.

**USPS example:** To test our approach on real-world data, we also applied the algorithm to the USPS database of 256-dimensional handwritten digits. For each of the ten digits, we randomly chose 300 examples from the training set and 50 examples from the test set. We used (10) and Gaussian kernels with $c = 0.50$, equaling twice the average of the data's variance in each dimensions. In figure 4, we give two possible depictions of

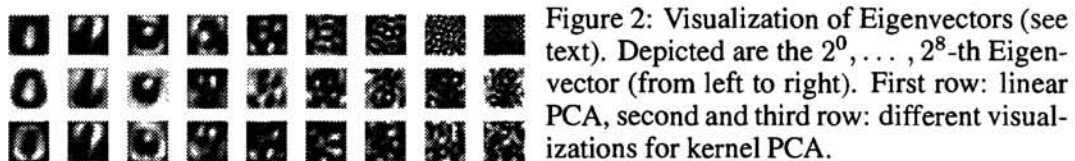

Figure 2: Visualization of Eigenvectors (see text). Depicted are the $2^0, \ldots, 2^8$-th Eigenvector (from left to right). First row: linear PCA, second and third row: different visualizations for kernel PCA.

the Eigenvectors found by kernel PCA, compared to those found by linear PCA for the USPS set. The second row shows the approximate pre-images of the Eigenvectors $V^k$, $k = 2^0, \ldots, 2^8$, found by our algorithm. In the third row each image is computed as follows: Pixel $i$ is the projection of the $\Phi$-image of the $i$-th canonical basis vector in input space onto the corresponding Eigenvector in features space (upper left $\Phi(e_1) \cdot V^k$, lower right $\Phi(e_{256}) \cdot V^k$). In the linear case, both methods would simply yield the Eigenvectors of linear PCA depicted in the first row; in this sense, they may be considered as generalized Eigenvectors in input space. We see that the first Eigenvectors are almost identical (except for signs). But we also see, that Eigenvectors in linear PCA start to concentrate on high-frequency structures already at smaller Eigenvalue size. To understand this, note that in linear PCA we only have a maximum number of 256 Eigenvectors, contrary to kernel PCA which gives us the number of training examples (here 3000) possible Eigenvectors. This

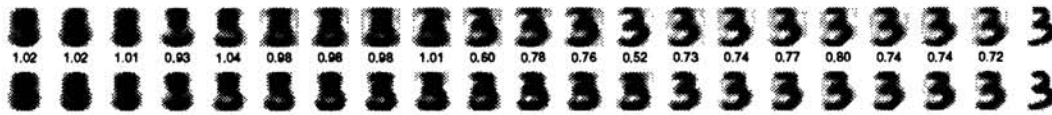

Figure 3: Reconstruction of USPS data. Depicted are the reconstructions of the first digit in the test set (original in last column) from the first $n = 1, \ldots, 20$ components for linear PCA (first row) and kernel PCA (second row) case. The numbers in between denote the fraction of squared distance measured towards the original example. For a small number of components both algorithms do nearly the same. For more components, we see that linear PCA yields a result resembling the original digit, whereas kernel PCA gives a result resembling a more prototypical 'three'

also explains some of the results we found when working with the USPS set. In these experiments, linear and kernel PCA were trained with the original data. Then we added (i) additive Gaussian noise with zero mean and standard deviation $\sigma = 0.5$ or (ii) 'speckle' noise with probability $p = 0.4$ (i.e. each pixel flips to black or white with probability $p/2$) to the test set. For the noisy test sets we computed the projections onto the first $n$ linear and nonlinear components, and carried out reconstruction for each case. The results were compared by taking the mean squared distance of each reconstructed digit from the noisy test set to its original counterpart. As a third experiment we did the same for the original test set (hence doing reconstruction, not de-noising). In the latter case, where the task is to reconstruct a given example as exactly as possible, linear PCA did better, at least when using more than about 10 components (figure 3). This is due to the fact that linear PCA starts earlier to account for fine structures, but at the same time it starts to reconstruct noise, as we will see in figure 4. Kernel PCA, on the other hand, yields recognizable results even for a small number of components, representing a prototype of the desired example. This is one reason why our approach did better than linear PCA for the de-noising example (figure 4). Taking the mean squared distance measured over the whole test set for the optimal number of components in linear and kernel PCA, our approach did better by a factor of 1.6 for the Gaussian noise, and 1.2 times better for the 'speckle' noise (the optimal number of components were 32 in linear PCA, and 512 and 256 in kernel PCA, respectively). Taking identical numbers of components in both algorithms, kernel PCA becomes up to 8 (!) times better than linear PCA. However, note that kernel PCA comes with a higher computational complexity.

## 5 Discussion

We have studied the problem of finding approximate pre-images of vectors in feature space, and proposed an algorithm to solve it. The algorithm can be applied to both reconstruction and de-noising. In the former case, results were comparable to linear PCA, while in the latter case, we obtained significantly better results. Our interpretation of this finding is as follows. Linear PCA can extract at most $N$ components, where $N$ is the dimensionality of the data. Being a basis transform, all $N$ components together fully describe the data. If the data are noisy, this implies that a certain fraction of the components will be devoted to the extraction of noise. Kernel PCA, on the other hand, allows the extraction of up to $\ell$ features, where $\ell$ is the number of training examples. Accordingly, kernel PCA can provide a larger number of features carrying information about the structure in the data (in our experiments, we had $\ell > N$). In addition, if the structure to be extracted is nonlinear, then linear PCA must necessarily fail, as we have illustrated with toy examples.

These methods, along with depictions of pre-images of vectors in feature space, provide some understanding of kernel methods which have recently attracted increasing attention. Open questions include (i) what kind of results kernels other than Gaussians will provide,

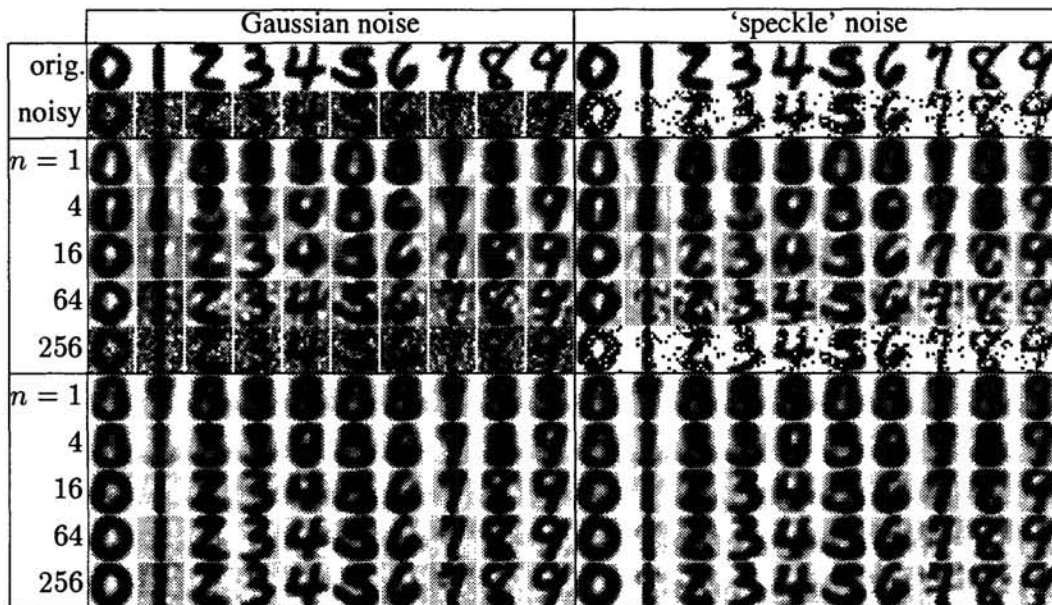

Figure 4: De-Noising of USPS data (see text). The left half shows: *top:* the first occurrence of each digit in the test set, *second row:* the upper digit with additive Gaussian noise ($\sigma = 0.5$), *following five rows:* the reconstruction for linear PCA using $n = 1, 4, 16, 64, 256$ components, and, *last five rows:* the results of our approach using the same number of components. In the right half we show the same but for 'speckle' noise with probability $p = 0.4$.

(ii) whether there is a more efficient way to solve either (6) or (8), and (iii) the comparison (and connection) to alternative nonlinear de-noising methods (cf. [5]).

## Footnotes

[1]For simplicity, we assume that the mapped data are centered in **F**. Otherwise, we have to go through the same algebra using $\tilde{\Phi}(\boldsymbol{x}) := \Phi(\boldsymbol{x}) - \langle \Phi(\boldsymbol{x}_i) \rangle$.

[2]If the kernel allows reconstruction of the dot–product in input space, and under the assumption that a pre–image exists, it is possible to construct it explicitly (cf. [7]). But clearly, these conditions do not hold true in general.

## References

[1] B. Boser, I. Guyon, and V.N. Vapnik. A training algorithm for optimal margin classifiers. In D. Haussler, editor, *Proc. COLT*, pages 144–152, Pittsburgh, 1992. ACM Press.

[2] C.J.C. Burges. Simplified support vector decision rules. In L. Saitta, editor, *Prooceedings, 13th ICML*, pages 71–77, San Mateo, CA, 1996.

[3] K.I. Diamantaras and S.Y. Kung. *Principal Component Neural Networks*. Wiley, New York, 1996.

[4] T. Hastie and W. Stuetzle. Principal curves. *JASA*, 84:502–516, 1989.

[5] S. Mallat and Z. Zhang. Matching Pursuits with time-frequency dictionaries. *IEEE Transactions on Signal Processing*, 41(12):3397–3415, December 1993.

[6] S. Saitoh. *Theory of Reproducing Kernels and its Applications*. Longman Scientific & Technical, Harlow, England, 1988.

[7] B. Schölkopf. *Support vector learning*. Oldenbourg Verlag, Munich, 1997.

[8] B. Schölkopf, P. Knirsch, A. Smola, and C. Burges. Fast approximation of support vector kernel expansions, and an interpretation of clustering as approximation in feature spaces. In P. Levi et. al., editor, *DAGM'98*, pages 124 – 132, Berlin, 1998. Springer.

[9] B. Schölkopf, A.J. Smola, and K.-R. Müller. Nonlinear component analysis as a kernel eigenvalue problem. *Neural Computation*, 10:1299–1319, 1998.
